# Boosting the Area Under the ROC Curve

**Philip M. Long**
plong@google.com

**Rocco A. Servedio**
rocco@cs.columbia.edu

## Abstract

We show that any weak ranker that can achieve an area under the ROC curve slightly better than $1/2$ (which can be achieved by random guessing) can be efficiently boosted to achieve an area under the ROC curve arbitrarily close to 1. We further show that this boosting can be performed even in the presence of independent misclassification noise, given access to a noise-tolerant weak ranker.

## 1 Introduction

**Background.** Machine learning is often used to identify members of a given class from a list of candidates. This can be formulated as a ranking problem, where the algorithm takes a input a list of examples of members and non-members of the class, and outputs a function that can be used to rank candidates. The goal is to have the top of the list enriched for members of the class of interest.

ROC curves [12, 3] are often used to evaluate the quality of a ranking function. A point on an ROC curve is obtained by cutting off the ranked list, and checking how many items above the cutoff are members of the target class ("true positives"), and how many are not ("false positives").

The AUC [1, 10, 3] (area under the ROC curve) is often used as a summary statistic. It is obtained by rescaling the axes so the true positives and false positives vary between $0$ and $1$, and, as the name implies, examining the area under the resulting curve.

The AUC measures the ability of a ranker to identify regions in feature space that are unusually densely populated with members of a given class. A ranker can succeed according to this criterion even if positive examples are less dense than negative examples everywhere, but, in order to succeed, it must identify where the positive examples tend to be. This is in contrast with classification, where, if $\mathbf{Pr}[y = 1|x]$ is less than $1/2$ everywhere, just predicting $y = -1$ everywhere would suffice.

**Our Results.** It is not hard to see that an AUC of $1/2$ can be achieved by random guessing (see [3]), thus it is natural to define a "weak ranker" to be an algorithm that can achieve AUC slightly above $1/2$. We show that any weak ranker can be boosted to a strong ranker that achieves AUC arbitrarily close to the best possible value of $1$.

We also consider the standard independent classification noise model, in which the label of each example is flipped with probability $\eta$. We show that in this setting, given a *noise-tolerant* weak ranker (that achieves nontrivial AUC in the presence of noisy data as described above), we can boost to a strong ranker that achieves AUC at least $1 - \epsilon$, for any $\eta < 1/2$ and any $\epsilon > 0$.

**Related work.** Freund, Iyer, Schapire and Singer [4] introduced RankBoost, which performs ranking with more fine-grained control over preferences between pairs of items than we consider here. They performed an analysis that implies a bound on the AUC of the boosted ranking function in terms of a different measure of the quality of weak rankers. Cortes and Mohri [2] theoretically analyzed the "typical" relationship between the error rate of a classifier based on thresholding a scoring function and the AUC obtained through the scoring function; they also pointed out the close relationship between the loss function optimized by RankBoost and the AUC. Rudin, Cortes, Mohri, and Schapire [11] showed that, when each of two classes are equally likely, the loss function optimized by AdaBoost coincides with the loss function of RankBoost. Noise-tolerant boosting has previously been studied for classification. Kalai and Servedio [7] showed that, if data is corrupted

with noise at a rate $\eta$, it is possible to boost the accuracy of any noise-tolerant weak learner arbitrarily close to $1 - \eta$, and they showed that it is impossible to boost beyond $1 - \eta$. In contrast, we show that, in the presence of noise at a rate arbitrarily close to $1/2$, the AUC can be boosted arbitrarily close to 1. Our noise tolerant boosting algorithm uses as a subroutine the "martingale booster" for classification of Long and Servedio [9].

**Methods.** The key observation is that a weak ranker can be used to find a "two-sided" weak classifier (Lemma 4), which achieves accuracy slightly better than random guessing on both positive and negative examples. Two-sided weak classifiers can be boosted to obtain accuracy arbitrarily close to 1, also on both the positive examples and the negative examples; a proof of this is implicit in the analysis of [9]. Such a two-sided strong classifier is easily seen to lead to AUC close to 1.

Why is it possible to boost past the AUC past the noise rate, when this is provably not possible for classification? Known approaches to noise-tolerant boosting [7, 9] force the weak learner to provide a two-sided weak hypothesis by balancing the distributions that are constructed so that both classes are equally likely. However, this balancing skews the distributions so that it is no longer the case that the event that an example is corrupted with noise is independent of the instance; randomization was used to patch this up in [7, 9], and the necessary slack was only available if the desired accuracy was coarser than the noise rate. (We note that the lower bound from [7] is proved using a construction in which the class probability of positive examples is less than the noise rate; the essence of that proof is to show that in that situation it is impossible to balance the distribution given access to noisy examples.) In contrast, having a weak ranker provides enough leverage to yield a two-sided weak classifier without needing any rebalancing.

**Outline.** Section 2 gives some definitions. In Section 3, we analyze boosting the AUC when there is no noise in an abstract model where the weak learner is given a distribution and returns a weak ranker, and sampling issues are abstracted away. In Section 4, we consider boosting in the presence of noise in a similarly abstract model. We address sampling issues in Section 5.

## 2    Preliminaries

**Rankings and AUC.** Throughout this work we let $X$ be a domain, $c : X \to \{-1, 1\}$ be a classifier, and $\mathcal{D}$ be a probability distribution over labeled examples $(x, c(x))$. We say that $\mathcal{D}$ is *nontrivial* (for $c$) if $\mathcal{D}$ assigns nonzero probability to both positive and negative examples. We write $\mathcal{D}^+$ to denote the marginal distribution over positive examples and $\mathcal{D}^-$ to denote the marginal distribution over negative examples, so $\mathcal{D}$ is a mixture of the distributions $\mathcal{D}^+$ and $\mathcal{D}^-$.

As has been previously pointed out, we may view any function $h : X \to \mathbf{R}$ as a ranking of $X$. Note that if $h(x_1) = h(x_2)$ then the ranking does not order $x_1$ relative to $x_2$. Given a ranking function $h : X \to \mathbf{R}$, for each value $\theta \in \mathbf{R}$ there is a point $(\alpha_\theta, \beta_\theta)$ on the *ROC curve of $h$*, where $\alpha_\theta$ is the false positive rate and $\beta_\theta$ is the true positive rate of the classifier obtained by thresholding $h$ at $\theta$: $\alpha_\theta = \mathcal{D}^-[h(x) \geq \theta]$ and $\beta_\theta = \mathcal{D}^+[h(x) \geq \theta]$. Every ROC curve contains the points $(0, 0)$ and $(1, 1)$ corresponding to $\theta = \infty$ and $-\infty$ respectively.

Given $h : X \to \mathbf{R}$ and $\mathcal{D}$, the AUC can be defined as $\mathrm{AUC}(h; \mathcal{D}) = \mathbf{Pr}_{u \in \mathcal{D}^+, v \in \mathcal{D}^-}[h(u) > h(v)] + \frac{1}{2}\mathbf{Pr}_{u \in \mathcal{D}^+, v \in \mathcal{D}^-}[h(u) = h(v)]$. It is well known (see e.g. [2, 6]) that the AUC as defined above is equal to the area under the ROC curve for $h$.

**Weak Rankers.** Fix any distribution $\mathcal{D}$. It is easy to see that any constant function $h$ achieves $\mathrm{AUC}(h; \mathcal{D}) = \frac{1}{2}$, and also that for $X$ finite and $\pi$ a random permutation of $X$, the expected AUC of $h(\pi(\cdot))$ is $\frac{1}{2}$ for any function $h$. This motivates the following definition:

**Definition 1** *A* weak ranker with advantage $\gamma$ *is an algorithm that, given any nontrivial distribution $\mathcal{D}$, returns a function $h : X \to \mathbf{R}$ that has $\mathrm{AUC}(h; \mathcal{D}) \geq \frac{1}{2} + \gamma$.*

In the rest of the paper we show how boosting algorithms originally designed for classification can be adapted to convert weak rankers into "strong" rankers (that achieve AUC at least $1 - \epsilon$) in a range of different settings.

# 3   From weak to strong AUC

The main result of this section is a simple proof that the AUC can be boosted. We achieve this in a relatively straightforward way by using the standard AdaBoost algorithm for boosting classifiers.

As in previous work [9], to keep the focus on the main ideas we will use an abstract model in which the booster successively passes distributions $\mathcal{D}_1, \mathcal{D}_2, ...$ to a weak ranker which returns ranking functions $h_1, h_2, ....$. When the original distribution $\mathcal{D}$ is uniform over a training set, as in the usual analysis of AdaBoost, this is easy to do. In this model we prove the following:

**Theorem 2** *There is an algorithm AUCBoost that, given access to a weak ranker with advantage $\gamma$ as an oracle, for any nontrivial distribution $\mathcal{D}$, outputs a ranking function with* AUC *at least $1 - \epsilon$. The AUCBoost algorithm makes $T = O(\frac{\log(1/\epsilon)}{\gamma^2})$ many calls to the weak ranker. If $\mathcal{D}$ has finite support of size $m$, AUCBoost takes $O(mT \log m)$ time.*

As can be seen from the observation that it does not depend on the relative frequency of positive and negative examples, the AUC requires a learner to perform well on both positive and negative examples. When such a requirement is imposed on a base classifier, it has been called *two-sided weak learning*. The key to boosting the AUC is the observation (Lemma 4 below) that a weak ranker can be used to generate a two-sided weak learner.

**Definition 3** *A $\gamma$ two-sided weak learner is an algorithm that, given a nontrivial distribution $\mathcal{D}$, outputs a hypothesis $h$ that satisfies both $\mathbf{Pr}_{x \in \mathcal{D}^+}[h(x) = 1] \geq \frac{1}{2} + \gamma$ and $\mathbf{Pr}_{x \in \mathcal{D}^-}[h(x) = -1] \geq \frac{1}{2} + \gamma$. We say that such an $h$ has* two-sided advantage $\gamma$ *with respect to $\mathcal{D}$.*

**Lemma 4** *Let $A$ be a weak ranking algorithm with advantage $\gamma$. Then there is a $\gamma/4$ two-sided weak learner $A'$ based on $A$ that always returns classifiers with equal error rate on positive and negative examples.*

**Proof:** Algorithm $A'$ first runs $A$ to get a real-valued ranking function $h : X \to \mathbf{R}$. Consider the ROC curve corresponding to $h$. Since the AUC is at least $\frac{1}{2} + \gamma$, there must be some point $(u, v)$ on the curve such that $v \geq u + \gamma$. Recall that, by the definition of the ROC curve, this means that there is a threshold $\theta$ such that $\mathcal{D}^+[h(x) \geq \theta] \geq \mathcal{D}^-[h(x) \geq \theta] + \gamma$. Thus, for the classifier obtained by thresholding $h$ at $\theta$, the class conditional error rates $p_+ \stackrel{\text{def}}{=} \mathcal{D}^+[h(x) < \theta]$ and $p_- \stackrel{\text{def}}{=} \mathcal{D}^-[h(x) \geq \theta]$ satisfy $p_+ + p_- \leq 1 - \gamma$. This in turn means that either $p_+ \leq \frac{1}{2} - \frac{\gamma}{2}$ or $p_- \leq \frac{1}{2} - \frac{\gamma}{2}$.

Suppose that $p_- \leq p_+$, so that $p_- \leq \frac{1}{2} - \frac{\gamma}{2}$ (the other case can be handled symmetrically). Consider the randomized classifier $g$ that behaves as follows: given input $x$, (a) if $h(x) < \theta$, it flips a biased coin, and with probability $\zeta \geq 0$, predicts 1, and with probability $1 - \zeta$, predicts $-1$, and (b) if $h(x) \geq \theta$, it predicts 1. Let $g(x, r)$ be the output of $g$ on input $x$ and with randomization $r$ and let $\epsilon_- \stackrel{\text{def}}{=} \mathbf{Pr}_{x \in \mathcal{D}^-, r}[g(x, r) = 1]$ and $\epsilon_+ \stackrel{\text{def}}{=} \mathbf{Pr}_{x \in \mathcal{D}^+, r}[g(x, r) = -1]$. We have $\epsilon_+ = (1 - \zeta)p_+$ and $\epsilon_- = p_- + \zeta(1 - p_-)$. Let us choose $\zeta$ so that $\epsilon_- = \epsilon_+$; that is, we choose $\zeta = \frac{p_+ - p_-}{1 + p_+ - p_-}$. This yields

$$\epsilon_- = \epsilon_+ = \frac{p_+}{1 + p_+ - p_-}. \tag{1}$$

For any fixed value of $p_-$ the RHS of (1) increases with $p_+$. Recalling that we have $p_+ + p_- \leq 1 - \gamma$, the maximum of (1) is achieved at $p_+ = 1 - \gamma - p_-$, in which case we have (defining $\epsilon \stackrel{\text{def}}{=} \epsilon_- = \epsilon_+$) $\epsilon = \frac{(1-\gamma) - p_-}{1 + (1-\gamma-p_-) - p_-} = \frac{(1-\gamma) - p_-}{2 - \gamma - 2p_-}$. The RHS of this expression is nonincreasing in $p_-$, and therefore is maximized at $p_-$ is 0, when it takes the value $\frac{1}{2} - \frac{\gamma}{2(2-\gamma)} \leq \frac{1}{2} - \frac{\gamma}{4}$. This completes the proof. $\square$

Figure 1 gives an illustration of the proof of the previous lemma; since the $y$-coordinate of (a) is at least $\gamma$ more than the $x$-coordinate and (b) lies closer to (a) than to $(1, 1)$, the $y$-coordinate of (b) is at least $\gamma/2$ more than the $x$-coordinate, which means that the advantage is at least $\gamma/4$.

We will also need the following simple lemma which shows that a classifier that is good on both the positive and the negative examples, when viewed as a ranking function, achieves a good AUC.

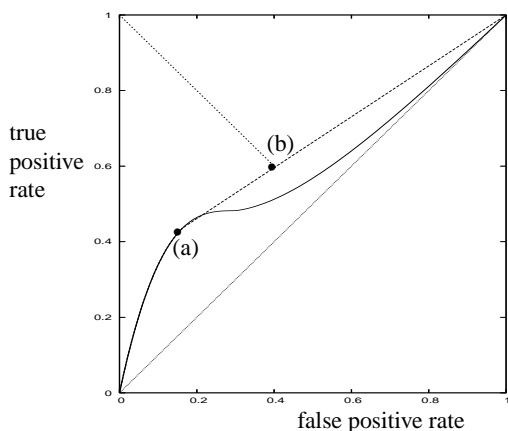

true positive rate

false positive rate

Figure 1: The curved line represents the ROC curve for ranking function $h$. The lower black dot (a) corresponds to the value $\theta$ and is located at $(p_-, 1-p_+)$. The straight line connecting $(0,0)$ and $(1,1)$, which corresponds to a completely random ranking, is given for reference. The dashed line (covered by the solid line for $0 \leq x \leq .16$) represents the ROC curve for a ranker $h'$ which agrees with $h$ on those $x$ for which $h(x) \geq \theta$ but randomly ranks those $x$ for which $h(x) < \theta$. The upper black dot (b) is at the point of intersection between the ROC curve for $h'$ and the line $y = 1-x$; its coordinates are $(\epsilon, 1 - \epsilon)$. The randomized classifier $g$ is equivalent to thresholding $h'$ with a value $\theta'$ corresponding to this point.

**Lemma 5** *Let* $h : X \to \{-1, 1\}$ *and suppose that* $\mathbf{Pr}_{x \in \mathcal{D}^+}[h(x) = 1] = 1 - \epsilon_+$ *and* $\mathbf{Pr}_{x \in \mathcal{D}^-}[h(x) = -1] = 1 - \epsilon_-$. *Then we have* $\mathrm{AUC}(h; \mathcal{D}) = 1 - \frac{\epsilon_+ + \epsilon_-}{2}$.

**Proof:** We have

$$\mathrm{AUC}(h; \mathcal{D}) = (1 - \epsilon_+)(1 - \epsilon_-) + \frac{\epsilon_+(1 - \epsilon_-) + \epsilon_-(1 - \epsilon_+)}{2} = 1 - \frac{\epsilon_+ + \epsilon_-}{2}. \qquad \square$$

**Proof of Theorem 2:** AUCBoost works by running AdaBoost on $\frac{1}{2}\mathcal{D}^+ + \frac{1}{2}\mathcal{D}^-$. In round $t$, each copy of AdaBoost passes its reweighted distribution $\mathcal{D}_t$ to the weak ranker, and then uses the process of Lemma 4 to convert the resulting weak ranking function to a classifier $h_t$ with two-sided advantage $\gamma/4$. Since $h_t$ has two-sided advantage $\gamma/4$, no matter how $\mathcal{D}_t$ decomposes into a mixture of $\mathcal{D}_t^+$ and $\mathcal{D}_t^-$, it must be the case that $\mathbf{Pr}_{(x,y) \in \mathcal{D}_t}[h_t(x) \neq y] \leq \frac{1}{2} - \gamma/4$.

The analysis of AdaBoost (see [5]) shows that $T = O\left(\frac{\log(1/\epsilon)}{\gamma^2}\right)$ rounds are sufficient for $H$ to have error rate at most $\epsilon$ under $\frac{1}{2}\mathcal{D}_+ + \frac{1}{2}\mathcal{D}_-$. Lemma 5 now gives that the classifier $H(x)$ is a ranking function with AUC at least $1 - \epsilon$.

For the final assertion of the theorem, note that at each round, in order to find the value of $\theta$ that defines $h_t$ the algorithm needs to minimize the sum of the error rates on the positive and negative examples. This can be done by sorting the examples using the weak ranking function (in $O(m \log m)$ time steps) and processing the examples in the resulting order, keeping running counts of the number of errors of each type. $\qquad \square$

## 4    Boosting weak rankers in the presence of misclassification noise

**The noise model: independent misclassification noise.** The model of *independent misclassification noise* has been widely studied in computational learning theory. In this framework there is a noise rate $\eta < 1/2$, and each example (positive or negative) drawn from distribution $\mathcal{D}$ has its true label $c(x)$ independently flipped with probability $\eta$ before it is given to the learner. We write $\mathcal{D}^\eta$ to denote the resulting distribution over (noise-corrupted) labeled examples $(x, y)$.

**Boosting weak rankers in the presence of independent misclassification noise.** We now show how the AUC can be boosted arbitrarily close to 1 even if the data given to the booster is corrupted with independent misclassification noise, using weak rankers that are able to tolerate independent misclassification noise. We note that this is in contrast with known results for boosting the accuracy of binary classifiers in the presence of noise; Kalai and Servedio [7] show that no "black-box" boosting algorithm can be guaranteed to boost the accuracy of an arbitrary noise-tolerant weak learner to accuracy $1 - \eta$ in the presence of independent misclassification noise at rate $\eta$.

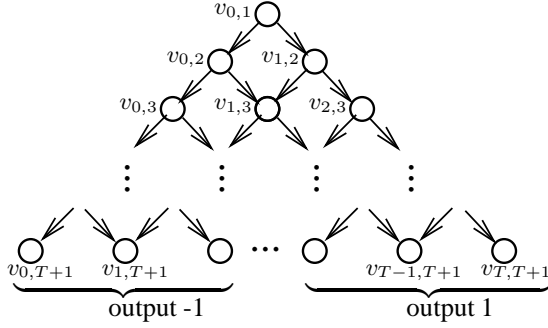

Figure 2: The branching program produced by the boosting algorithm. Each node $v_{i,t}$ is labeled with a weak classifier $h_{i,t}$; left edges correspond to -1 and right edges to 1.

As in the previous section we begin by abstracting away sampling issues and using a model in which the booster passes a distribution to a weak ranker. Sampling issues will be treated in Section 5.

**Definition 6** *A* noise-tolerant weak ranker with advantage $\gamma$ *is an algorithm with the following property: for any noise rate $\eta < 1/2$, given a noisy distribution $\mathcal{D}^\eta$, the algorithm outputs a ranking function $h : X \to \mathbf{R}$ such that $\mathrm{AUC}(h; \mathcal{D}) \geq \frac{1}{2} + \gamma$.*

Our algorithm for boosting the AUC in the presence of noise uses the Basic MartiBoost algorithm (see Section 4 of [9]). This algorithm boosts any two-sided weak learner to arbitrarily high accuracy and works in a series of rounds. Before round $t$ the space of labeled examples is partitioned into a series of bins $B_{0,t}, ..., B_{t-1,t}$. (The original bin $B_{0,1}$ consists of the entire space.) In the $t$-th round the algorithm first constructs distributions $\mathcal{D}_{0,t}, ..., \mathcal{D}_{t-1,t}$ by conditioning the original distribution $\mathcal{D}$ on membership in $B_{0,t}, ..., B_{t-1,t}$ respectively. It then calls a two-sided weak learner $t$ times using each of $\mathcal{D}_{0,t}, ..., \mathcal{D}_{t-1,t}$, getting weak classifiers $h_{0,t}, ..., h_{t-1,t}$ respectively. Having done this, it creates $t + 1$ bins for the next round by assigning each element $(x, y)$ of $B_{i,t}$ to $B_{i,t+1}$ if $h_{i,t}(x) = -1$ and to $B_{i+1,t+1}$ otherwise. Training proceeds in this way for a given number $T$ of rounds, which is an input parameter of the algorithm.

The output of Basic MartiBoost is a layered branching program defined as follows. There is a node $v_{i,t}$ for each round $1 \leq t \leq T + 1$ and each index $0 \leq i < t$ (that is, for each bin constructed during training). An item $x$ is routed through the branching program the same way a labeled example $(x, y)$ would have been routed during the training phase: it starts in node $v_{0,1}$, and from each node $v_{i,t}$ it goes to $v_{i,t+1}$ if $h_{i,t}(x) = -1$, and to $v_{i+1,t+1}$ otherwise. When the item $x$ arrives at a terminal node of the branching program in layer $T + 1$, it is at some node $v_{j,T+1}$. The prediction is 1 if $j \geq T/2$ and is $-1$ if $j < T/2$; in other words, the prediction is according to the majority vote of the weak classifiers that were encountered along the path through the branching program that the example followed. See Figure 3.

The following lemma is proved in [9]. (The crux of the proof is the observation that positive (respectively, negative) examples are routed through the branching program according to a random walk that is biased to the right (respectively, left); hence the name "martingale boosting.")

**Lemma 7 ([9])** *Suppose that Basic MartiBoost is provided with a hypothesis $h_{i,t}$ with two-sided advantage $\gamma$ w.r.t. $\mathcal{D}_{i,t}$ at each node $v_{i,t}$. Then for $T = O(\log(1/\epsilon)/\gamma^2)$, Basic MartiBoost constructs a branching program $H$ such that $\mathcal{D}^+[H(x) = -1] \leq \epsilon$ and $\mathcal{D}^-[H(x) = 1] \leq \epsilon$.*

We now describe our noise-tolerant AUC boosting algorithm, which we call Basic MartiRank. Given access to a noise-tolerant weak ranker $A$ with advantage $\gamma$, at each node $v_{i,t}$ the Basic MartiRank algorithm runs $A$ and proceeds as described in Lemma 4 to obtain a weak classifier $h_{i,t}$. Basic MartiRank runs Basic MartiBoost with $T = O(\log(1/\epsilon)/\gamma^2)$ and simply uses the resulting classifier $H$ as its ranking function. The following theorem shows that Basic MartiRank is an effective AUC booster in the presence of independent misclassification noise:

**Theorem 8** *Fix any $\eta < 1/2$ and any $\epsilon > 0$. Given access to $\mathcal{D}^\eta$ and a noise-tolerant weak ranker $A$ with advantage $\gamma$, Basic MartiRank outputs a branching program $H$ such that $\mathrm{AUC}(H; \mathcal{D}) \geq 1 - \epsilon$.*

**Proof:** Fix any node $v_{i,t}$ in the branching program. The crux of the proof is the following simple observation: for a labeled example $(x, y)$, the route through the branching program that is taken

by $(x, y)$ is determined completely by the predictions of the base classifiers, i.e. only by $x$, and is unaffected by the value of $y$. Consequently if $\mathcal{D}_{i,t}$ denotes the original noiseless distribution $\mathcal{D}$ conditioned on reaching $v_{i,t}$, then the noisy distribution conditioned on reaching $v_{i,t}$, i.e. $(\mathcal{D}^\eta)_{i,t}$, is simply $\mathcal{D}_{i,t}$ corrupted with independent misclassification noise, i.e. $(\mathcal{D}_{i,t})^\eta$. So each time the noise-tolerant weak ranker $A$ is invoked at a node $v_{i,t}$, it is indeed the case that the distribution that it is given is an independent misclassification noise distribution. Consequently $A$ does construct weak rankers with AUC at least $1/2 + \gamma$, and the conversion of Lemma 4 yields weak classifiers that have advantage $\gamma/4$ with respect to the underlying distribution $\mathcal{D}_{i,t}$. Given this, Lemma 7 implies that the final classifier $H$ has error at most $\epsilon$ on both positive and negative examples drawn from the original distribution $\mathcal{D}$, and Lemma 5 then implies that $H$, viewed a ranker, achieves AUC at least $1 - \epsilon$. $\square$

In [9], a more complex variant of Basic MartiBoost, called Noise-Tolerant SMartiBoost, is presented and is shown to boost any noise-tolerant weak learning algorithm to any accuracy less than $1 - \eta$ in the presence of independent misclassification noise. In contrast, here we are using just the Basic MartiBoost algorithm itself, and can achieve any AUC value $1 - \epsilon$ even for $\epsilon < \eta$.

## 5    Implementing MartiRank with a distribution oracle

In this section we analyze learning from random examples. Formally, we assume that the weak ranker is given access to an oracle for the noisy distribution $\mathcal{D}^\eta$. We thus now view a *noise-tolerant weak ranker with advantage $\gamma$* as an algorithm $A$ with the following property: for any noise rate $\eta < 1/2$, given access to an oracle for $\mathcal{D}^\eta$, the algorithm outputs a ranking function $h : X \to \mathbf{R}$ such that $\text{AUC}(h; \mathcal{D}) \geq \frac{1}{2} + \gamma$.

We let $m_A$ denote the number of examples from each class that suffice for $A$ to construct a ranking function as described above. In other words, if $A$ is provided with a sample of draws from $\mathcal{D}^\eta$ such that each class, positive and negative, has at least $m_A$ points in the sample with that true label, then algorithm $A$ outputs a $\gamma$-advantage weak ranking function. (Note that for simplicity we are assuming here that the weak ranker always constructs a weak ranking function with the desired advantage, i.e. we gloss over the usual confidence parameter $\delta$; this can be handled with an entirely standard analysis.)

In order to achieve a computationally efficient algorithm in this setting we must change the Marti-Rank algorithm somewhat; we call the new variant Sampling Martirank, or SMartiRank. We prove that SMartiRank is computationally efficient, has moderate sample complexity, and efficiently generates a high-accuracy final ranking function with respect to the underlying distribution $\mathcal{D}$.

Our approach follows the same general lines as [9] where an oracle implementation is presented for the MartiBoost algorithm. The main challenge in [9] is the following: for each node $v_{i,t}$ in the branching program, the boosting algorithm considered there must simulate a balanced version of the induced distribution $\mathcal{D}_{i,t}$ which puts equal weight on positive and negative examples. If only a tiny fraction of examples drawn from $\mathcal{D}$ are (say) positive and reach $v_{i,t}$, then it is very inefficient to simulate this balanced distribution (and in a noisy scenario, as discussed earlier, if the noise rate is high relative to the frequency of the desired class then it may in fact be impossible to simulate the balanced distribution). The solution in [9] is to "freeze" any such node and simply classify any example that reaches it as negative; the analysis argues that since only a tiny fraction of positive examples reach such nodes, this freezing only mildly degrades the accuracy of the final hypothesis.

In the ranking scenario that we now consider, we do not need to construct balanced distributions, but we do need to obtain a non-negligible number of examples from each class in order to run the weak learner at a given node. So as in [9] we still freeze some nodes, but with a twist: we now freeze nodes which have the property that for some class label (positive or negative), only a tiny fraction of examples from $\mathcal{D}$ *with that class label* reach the node. With this criterion for freezing we can prove that the final classifier constructed has high accuracy both on positive and negative examples, which is what we need to achieve good AUC. We turn now to the details.

Given a node $v_{i,t}$ and a bit $b \in \{-1, 1\}$, let $p_{i,t}^b$ denote $\mathcal{D}[x \text{ reaches } v_{i,t} \text{ and } c(x) = b]$. The SMartiRank algorithm is like Basic MartiBoost but with the following difference: for each node $v_{i,t}$

and each value $b \in \{-1, 1\}$, if

$$p_{i,t}^b < \frac{\epsilon \cdot \mathcal{D}[c(x) = b]}{T(T+1)} \qquad (2)$$

then the node $v_{i,t}$ is "frozen," i.e. it is labeled with the bit $1 - b$ and is established as a terminal node with no outgoing edges. (If this condition holds for both values of $b$ at a particular node $v_{i,t}$ then the node is frozen and either output value may be used as the label.) The following theorem establishes that if SMartiRank is given weak classifiers with two-sided advantage at each node that is not frozen, it will construct a hypothesis with small error rate on both positive and negative examples:

**Theorem 9** *Suppose that the SMartiRank algorithm as described above is provided with a hypothesis $h_{i,t}$ that has two-sided advantage $\gamma$ with respect to $\mathcal{D}_{i,t}$ at each node $v_{i,t}$ that is not frozen. Then for $T = O(\log(1/\epsilon)/\gamma^2)$, the final branching program hypothesis $H$ that SMartiRank constructs will have $\mathcal{D}^+[H(x) = -1] \leq \epsilon$ and $\mathcal{D}^-[H(x) = 1] \leq \epsilon$.*

**Proof:** We analyze $\mathcal{D}^+[h(x) = -1]$; the other case is symmetric.

Given an unlabeled instance $x \in X$, we say that $x$ *freezes at node* $v_{i,t}$ if $x$'s path through the branching program causes it to terminate at a node $v_{i,t}$ with $t < T + 1$ (i.e. at a node $v_{i,t}$ which was frozen by SMartiRank). We have $\mathcal{D}[x \text{ freezes and } c(x) = 1] = \sum_{i,t} \mathcal{D}[x \text{ freezes at } v_{i,t} \text{ and } c(x) = 1] \leq \sum_{i,t} \frac{\epsilon \cdot \mathcal{D}[c(x)=1]}{T(T+1)} \leq \frac{\epsilon}{2} \cdot \mathcal{D}[c(x) = 1]$. Consequently we have

$$\mathcal{D}^+[x \text{ freezes}] = \frac{\mathcal{D}[x \text{ freezes and } c(x) = 1]}{\mathcal{D}[c(x) = 1]} < \frac{\epsilon}{2}. \qquad (3)$$

Naturally, $\mathcal{D}^+[h(x) = -1] = \mathcal{D}^+[(h(x) = -1) \& (x \text{ freezes})] + \mathcal{D}^+[(h(x) = -1) \& (x \text{ does not freeze})]$. By (3), this is at most $\frac{\epsilon}{2} + \mathcal{D}^+[(h(x) = -1) \& (x \text{ does not freeze})]$. Arguments identical to those in the last two paragraphs of the proof of Theorem 3 in [9] show that $\mathcal{D}^+[(h(x) = -1) \& (x \text{ does not freeze})] \leq \frac{\epsilon}{2}$, and we are done. □

We now describe how SMartiRank can be run given oracle access to $\mathcal{D}^\eta$ and sketch the analysis of the required sample complexity (some details are omitted because of space limits). For simplicity of presentation we shall assume that the booster is given the value $p \overset{\text{def}}{=} \min\{\mathcal{D}[c(x) = -1], \mathcal{D}[c(x) = 1]\}$; we note if that $p$ is not given *a priori*, a standard "guess and halve" technique can be used to efficiently obtain a value that is within a multiplicative factor of two of $p$, which is easily seen to suffice.    We also make the standard assumption (see [7, 9]) that the noise rate $\eta$ is known; this assumption can similarly be removed by having the algorithm "guess and check" the value to sufficiently fine granularity. Also, the confidence can be analyzed using the standard appeal to the union bound – details are omitted.

SMartiRank will replace (2) with a comparison of sample estimates of the two quantities. To allow for the fact that they are just estimates, it will be more conservative, and freeze when the estimate of $p_{i,t}^b$ is at most $\frac{\epsilon}{4T(T+1)}$ times the estimate of $D[c(x) = b]$.

We first observe that for any distribution $\mathcal{D}$ and any bit $b$, we have $\mathbf{Pr}_{(x,y) \sim \mathcal{D}^\eta}[y = b] = \eta + (1 - 2\eta)\mathbf{Pr}_{(x,c(x)) \sim \mathcal{D}}[c(x) = b]$, which is equivalent to $\mathcal{D}[c(x) = b] = \frac{\mathcal{D}^\eta[y=b]-\eta}{1-2\eta}$. Consequently, given an empirical estimate of $\mathcal{D}^\eta[y = b]$ that is accurate to within an additive $\pm \frac{p(1-2\eta)}{10}$ (which can easily be obtained from $O(\frac{1}{p^2(1-2\eta)^2})$ draws to $\mathcal{D}^\eta$), it is possible to estimate $\mathcal{D}[c(x) = b]$ to within an additive $\pm p/10$, and thus to estimate the RHS of (2) to within an additive $\pm \frac{\epsilon p}{10T(T+1)}$. Now in order to determine whether node $v_{i,t}$ should be frozen, we must compare this estimate with a similarly accurate estimate of $p_{i,t}^b$ (arguments similar to those of, e.g., Section 6.3 of [9] can be used to show that it suffices to run the algorithm using these estimated values). We have

$$
\begin{aligned}
p_{i,t}^b &= \mathcal{D}[x \text{ reaches } v_{i,t}] \cdot \mathcal{D}[c(x) = b \mid x \text{ reaches } v_{i,t}] = \mathcal{D}^\eta[x \text{ reaches } v_{i,t}] \cdot \mathcal{D}_{i,t}[c(x) = b] \\
&= \mathcal{D}^\eta[x \text{ reaches } v_{i,t}] \cdot \left( \frac{\mathcal{D}_{i,t}^\eta[y = b] - \eta}{1 - 2\eta} \right).
\end{aligned}
$$

A standard analysis (see e.g. Chapter 5 of [8]) shows that this quantity can be estimated to additive accuracy $\pm \tau$ using $\text{poly}(1/\tau, 1/(1-2\eta))$ many calls to $\mathcal{D}^\eta$ (briefly, if $\mathcal{D}^\eta[x \text{ reaches } v_{i,t}]$ is less than

$\tau(1-2\eta)$ then an estimate of 0 is good enough, while if it is greater than $\tau(1-2\eta)$ then a $\tau$-accurate estimate of the second multiplicand can be obtained using $O(\frac{1}{\tau^3(1-2\eta)^3})$ draws from $\mathcal{D}^\eta$, since at least a $\tau(1-2\eta)$ fraction of draws will reach $v_{i,t}$.) Thus for each $v_{i,t}$, we can determine whether to freeze it in the execution of SMartiRank using $\text{poly}(T, 1/\epsilon, 1/p, 1/(1-2\eta))$ draws from $\mathcal{D}^\eta$.

For each of the nodes that are not frozen, we must run the noise-tolerant weak ranker $A$ using the distribution $\mathcal{D}_{i,t}^\eta$. As discussed at the beginning of this section, this requires that we obtain a sample from $\mathcal{D}_{i,t}^\eta$ containing at least $m_A$ examples whose true label belongs to each class. The expected number of draws from $\mathcal{D}^\eta$ that must be made in order to receive an example from a given class is $1/p$, and since $v_{i,t}$ is not frozen, the expected number of draws from $\mathcal{D}^\eta$ belonging to a given class that must be made in order to simulate a draw from $D_{i,t}^\eta$ belonging to that class is $O(T^2/\epsilon)$. Thus, $O(T^2 m_A/(\epsilon p))$ many draws from $\mathcal{D}^\eta$ are required in order to run the weak learner $A$ at any particular node. Since there are $O(T^2)$ many nodes overall, we have that all in all $O(T^4 m_A/(\epsilon p))$ many draws from $\mathcal{D}^\eta$ are required, in addition to the $\text{poly}(T, 1/\epsilon, 1/p, 1/(1-2\eta))$ draws required to identify which nodes to freeze. Recalling that $T = O(\log(1/\epsilon)/\gamma^2)$, all in all we have:

**Theorem 10** *Let $\mathcal{D}$ be a nontrivial distribution over $X$, $p = \min\{\mathcal{D}[c(x) = -1], \mathcal{D}[c(x) = 1]\}$, and $\eta < \frac{1}{2}$. Given access to an oracle for $\mathcal{D}^\eta$ and a noise-tolerant weak ranker $A$ with advantage $\gamma$, the SMartiRank algorithm makes $m_A \cdot \text{poly}(\frac{1}{\epsilon}, \frac{1}{\gamma}, \frac{1}{1-2\eta}, \frac{1}{p})$ calls to $\mathcal{D}^\eta$, and and with probability $1 - \delta$ outputs a branching program $H$ such that $\text{AUC}(h; \mathcal{D}) \geq 1 - \epsilon$.*

## Acknowledgement

We are very grateful to Naoki Abe for suggesting the problem of boosting the AUC.

## References

[1] A. P. Bradley. Use of the area under the ROC curve in the evaluation of machine learning algorithms. *Pattern Recognition*, 30:1145–1159, 1997.

[2] C. Cortes and M. Mohri. AUC optimization vs. error rate minimzation. In *NIPS 2003*, 2003.

[3] T. Fawcett. ROC graphs: Notes and practical considerations for researchers. Technical Report HPL-2003-4, HP, 2003.

[4] Y. Freund, R. Iyer, R. E. Schapire, and Y. Singer. An efficient boosting algorithm for combining preferences. *Journal of Machine Learning Research*, 4(6):933–970, 2004.

[5] Y. Freund and R. Schapire. A decision-theoretic generalization of on-line learning and an application to boosting. *Journal of Computer and System Sciences*, 55(1):119–139, 1997.

[6] J. Hanley and B. McNeil. The meaning and use of the area under a receiver operating characteristic (ROC) curve. *Radiology*, 143(1):29–36, 1982.

[7] A. Kalai and R. Servedio. Boosting in the presence of noise. *Journal of Computer & System Sciences*, 71(3):266–290, 2005. Preliminary version in *Proc. STOC'03*.

[8] M. Kearns and U. Vazirani. *An introduction to computational learning theory*. MIT Press, Cambridge, MA, 1994.

[9] P. Long and R. Servedio. Martingale boosting. In *Proceedings of the Eighteenth Annual Conference on Computational Learning Theory (COLT)*, pages 79–94, 2005.

[10] F. Provost, T. Fawcett, and Ron Kohavi. The case against accuracy estimation for comparing induction algorithms. *ICML*, 1998.

[11] C. Rudin, C. Cortes, M. Mohri, and R. E. Schapire. Margin-based ranking meets boosting in the middle. *COLT*, 2005.

[12] J. A. Swets. *Signal detection theory and ROC analysis in psychology and diagnostics: Collected papers*. Lawrence Erlbaum Associates, 1995.
